# A Neural Model of Delusions and Hallucinations in Schizophrenia

**Eytan Ruppin and James A. Reggia**
Department of Computer Science
University of Maryland, College Park, MD 20742
ruppin@cs.umd.edu    reggia@cs.umd.edu

**David Horn**
School of Physics and Astronomy,
Tel Aviv University, Tel Aviv 69978, Israel
horn@vm.tau.ac.il

## Abstract

We implement and study a computational model of Stevens' [1992] theory of the pathogenesis of schizophrenia. This theory hypothesizes that the onset of schizophrenia is associated with reactive synaptic regeneration occurring in brain regions receiving degenerating temporal lobe projections. Concentrating on one such area, the frontal cortex, we model a frontal module as an associative memory neural network whose input synapses represent incoming temporal projections. We analyze how, in the face of weakened external input projections, compensatory strengthening of internal synaptic connections and increased noise levels can maintain memory capacities (which are generally preserved in schizophrenia). However, These compensatory changes adversely lead to spontaneous, biased retrieval of stored memories, which corresponds to the occurrence of schizophrenic delusions and hallucinations without any apparent external trigger, and for their tendency to concentrate on just few central themes. Our results explain why these symptoms tend to wane as schizophrenia progresses, and why delayed therapeutical intervention leads to a much slower response.

# 1   Introduction

There has been a growing interest in recent years in the use of neural models to investigate various brain pathologies and their cognitive and behavioral effects. Recent published examples of such studies include models of cortical plasticity following stroke, Alzheimer's disease and schizophrenia, and cognitive and behavioral explorations of aphasia, acquired dyslexia and affective disorders (reviewed in [1, 2]). Continuing this line of study, we present a computational account linking specific pathological synaptic changes that are postulated to occur in schizophrenia, and the emergence of schizophrenic delusions and hallucinations. The latter symptoms denote persistent, unrealistic, psychotic thoughts (delusions) or percepts (hallucinations) that may at times flood the patient in an overwhelming, stressful manner.

The wealth of data gathered concerning the pathophysiology of schizophrenia supports the involvement of both the frontal and the temporal lobes. On the one hand, there are atrophic changes in the hippocampus and parahippocampal areas including neuronal loss and gliosis. On the other hand, neurochemical and morphometric studies testify to an expansion of various receptor binding sites and increased dendritic branching in the frontal cortex of schizophrenics. Stevens has recently presented a theory linking these temporal and frontal findings, claiming that the onset of schizophrenia is associated with reactive anomalous sprouting and synaptic reorganization taking place in the projection sites of degenerating temporal neurons, including (among various cortical and subcortical structures) the frontal lobes [3].

This paper presents a computational study of Stevens' theory. Within the framework of a memory model of hippocampal-frontal interaction, we show that the introduction of the 'microscopic' synaptic changes that underlie Stevens' hypothesis can help preserve memory function but results in specific 'pathological' changes in the 'macroscopic' behavior of the network. A small subset of the patterns stored in the network are now spontaneously retrieved at times, without being cued by any specific input pattern. This emergent behavior shares some of the important characteristics of schizophrenic delusions and hallucinations, which frequently appear in the absence of any apparent external trigger, and tend to concentrate on a limited set of recurrent themes [4]. Memory capacities are fairly preserved in schizophrenics, until late stages of the disease [5]. In Section 2 we present our model. The analytical and numerical results obtained are described in Section 3, followed by our conclusions in Section 4.

# 2   The Model

As illustrated in Figure 1, we model a frontal module as an associative memory attractor neural network, receiving its input memory cues from decaying *external* input fibers (representing the degenerating temporal projections). The network's *internal* connections, which store the memorized patterns, undergo synaptic strengthening changes that model the reactive synaptic regeneration within the frontal module. The effect of other *diffuse* external projections is modeled as background noise. A frontal module represents a macro-columnar unit that has been suggested as a basic functional building block of the neocortex [6]. The assumption that memory retrieval from the frontal cortex is invoked by the firing of incoming

temporal projections is based on the notion that temporal structures have an important role in establishing long-term memory in the neocortex and in the retrieval of facts and events (e.g., [7]).

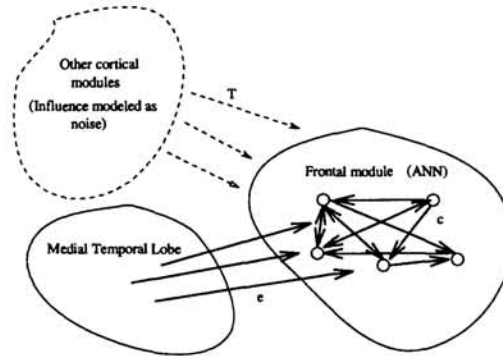

Figure 1: A schematic illustration of the model. A frontal module is modeled as an attractor neural network whose neurons receive inputs via three kinds of connections: internal connections from other frontal neurons, external connections from temporal lobe neurons, and diffuse external connections from other cortical modules, modeled as noise.

The attractor network we use is a biologically-motivated variant of Hopfield's ANN model, proposed by Tsodyks & Feigel'man [8]. Each neuron $i$ is described by a binary variable $S_i = \{1, 0\}$ denoting an active (firing) or passive (quiescent) state, respectively. $M = \alpha N$ distributed memory patterns $\xi^\mu$ are stored in the network. The elements of each memory pattern are chosen to be 1 (0) with probability $p$ $(1-p)$ respectively, with $p \ll 1$. All $N$ neurons in the network have a fixed uniform threshold $\theta$.

In its initial, undamaged state, the weights of the internal synaptic connections are

$$W_{ij} = \frac{c_0}{N} \sum_{\mu=1}^{M} (\xi^\mu{}_i - p)(\xi^\mu{}_j - p) \,, \tag{1}$$

where $c_0 = 1$. The post-synaptic potential (input field) $h_i$ of neuron $i$ is the sum of internal contributions from other neurons and external projections $F_i{}^e$

$$h_i(t) = \sum_j W_{ij} S_j(t-1) + F_i{}^e \,. \tag{2}$$

The updating rule for neuron $i$ at time $t$ is given by

$$S_i(t) = \begin{cases} 1, & \text{with prob. } G(h_i(t) - \theta) \\ 0, & \text{otherwise} \end{cases} \tag{3}$$

where $G$ is the sigmoid function $G(x) = 1/(1+\exp(-x/T))$, and $T$ denotes the noise level. The activation level of the stored memories is measured by their *overlaps $m^\mu$* with the current state of the network, defined by

$$m^\mu(t) = \frac{1}{p(1-p)N} \sum_{i=1}^{N} (\xi_i^\mu - p)S_i(t) \ . \tag{4}$$

*Stimulus-dependent retrieval* is modeled by orienting the field $F^e$ with one of the memorized patterns (the *cued* pattern, say $\xi^1$), such that

$$F_i^e = e \cdot \xi^1{}_i \ , \quad (e > 0) \ . \tag{5}$$

Following the presentation of an external input cue, the network state evolves until it converges to a stable state. The network parameters are tuned such that in its initial, undamaged state it correctly retrieves the cued patterns ($e_0 = 0.035$, $c_0 = 1$, $T = 0.005$).

We also examine the network's behavior *in the absence of any specific stimulus*. The network may either continue to wander around in a state of random low baseline activity, or it may converge onto a stored memory state. We refer to the latter process as *spontaneous retrieval*.

Our investigation of Stevens' work proceeds in two stages. First we examine and analyze the behavior of the network when it undergoes uniform synaptic changes that represent the pathological changes occurring in accordance with Stevens' theory. These include the weakening of external input projections ($e \downarrow$) and the increase in the internal projections ($c \uparrow$) and noise levels ($T \uparrow$). In the second stage, we add the assumption that the internal synaptic compensatory changes have an additional Hebbian activity-dependent component, and examine the effect of the rule

$$W_{ij}(t) = W_{ij}(t-1) + \frac{\gamma}{N}(\bar{S}_i - p)(\bar{S}_j - p) \ , \tag{6}$$

where $\bar{S}_k$ is 1 (0) only if neuron $k$ has been consecutively firing (quiescent) for the last $\tau$ iterations, and $\gamma$ is a constant.

## 3    Results

We now show some simulation and analytic results, examining the effects of the 'microscopic' pathological changes, taking place in accordance with Stevens' theory, on the 'macroscopic' behavior of the network. The analytical results presented have been derived by calculating the magnitude of randomly formed initial 'biases', and comparing their effect on the network's dynamics versus the effect of externally presented input cues. This comparison is performed by formulating a corresponding overlap master equation, whose fixed point dynamics are investigated via phase-plane analysis, as described in [9]. First, we study whether the reactive synaptic changes (occurring in both internal and external, diffuse synapses) are really compensatory, i.e., to what extent can they help maintaining memory capacities in the face of degenerating external input synapses. As illustrated in Figure 2, we find that increased noise levels can (up to some degree) preserve memory retrieval in the face of decreased external input strength. Increased synaptic strengthening preserves

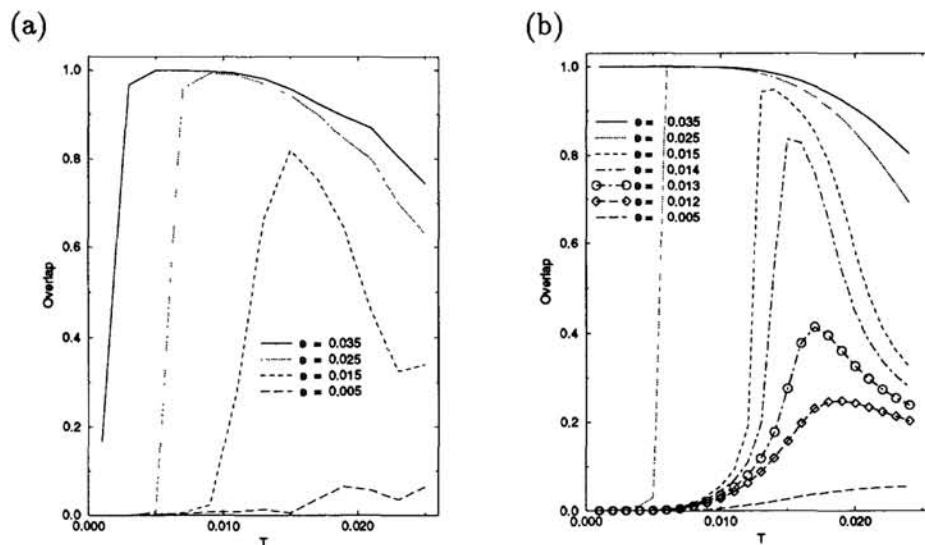

Figure 2: Stimulus-dependent retrieval performance, measured by the average final overlap $m$, as a function of the noise level $T$. Each curve displays this relation at a different magnitude of external input projections $e$. (a) Simulation results. (b) Analytic approximation.

memory retrieval in a similar manner, and the combined effect of these synaptic compensatory measures is synergistic.

Second, although the compensatory synaptic changes help maintain memory retrieval capacities, they necessarily have adverse effects, leading eventually to the emergence of spontaneous activation of non-cued memory patterns; the network converges to some of its memory patterns in a pathological, autonomous manner, in the absence of any external input stimuli. This emergence of pathological spontaneous retrieval, when either the noise level or the internal synaptic strength (or both) are increased beyond some point, is demonstrated in Figure 3.

Third, when the compensatory regeneration of internal synapses has an additional Hebbian component (representing a period of increased activity-dependent plasticity due to the regenerative synaptic changes), a *biased* spontaneous retrieval distribution is obtained. That is, as time evolves (measured in time units of 'trials'), the distribution of patterns spontaneously retrieved by the network in a pathological manner tends to concentrate only on one or two of all the memory patterns stored in the network, as is shown in Figure 4a. This highly peaked distribution is maintained for a few hundred additional trials until memory retrieval sharply collapses to zero as a global mixed-state attractor is formed. Such a mixed attractor state does not have very high overlap with any memorized pattern, and thus does not represent any well-defined cognitive or perceptual item. It is an end state of the Hebbian, activity-dependent evolution of the network. Yet, even after activity-dependent changes ensue, if spontaneous activity does not emerge the distribution of retrieved memories remains homogeneous (see Figure 4b). Eventually, a global

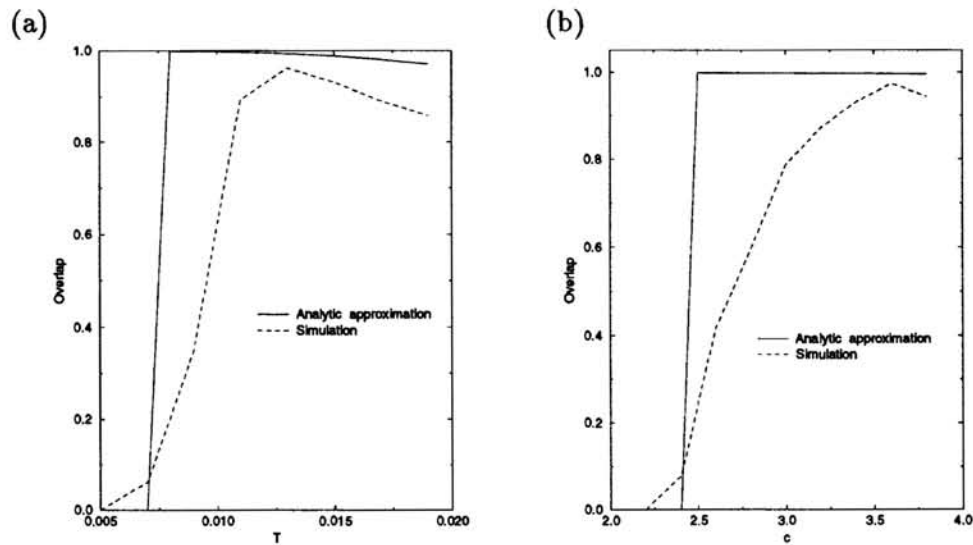

Figure 3: (a) Spontaneous retrieval, measured as the highest final overlap $m$ achieved with any of the stored memory patterns, displayed as a function of the noise level $T$. $c = 1$. (b) Spontaneous retrieval as a function of internal synaptic compensation factor $c$. $T = 0.009$.

mixed-state attractor is formed, and the network looses its retrieval capacities, but during this process no memory pattern gets to dominate the retrieval output. Our results remain qualitatively similar even when bounds are placed on the absolute magnitude of the synaptic weights.

## 4 Conclusions

Our results suggest that the formation of biased spontaneous retrieval requires the concomitant occurrence of both degenerative changes in the external input fibers, and regenerative Hebbian changes in the intra-modular synaptic connections. They add support to the plausibility of Stevens' theory by showing that it may be realized within a neural model, and account for a few characteristics of schizophrenic symptoms:

- The emergence of spontaneous, non-homogeneous retrieval is a self-limiting phenomenon (as eventually a cognitively meaningless global attractor is formed) - this parallels the clinical finding that as schizophrenia progresses both delusions and hallucinations tend to wane, while negative symptoms are enhanced [10].

- Once converged to, the network has a much larger tendency to remain in a biased memory state than in a non biased one - this is in accordance with the persistent characteristic of schizophrenic florid symptoms.

- As more spontaneous retrieval trials occur the frequency of spontaneous retrieval increases - indeed, while early treatment in young psychotic adults

(a) (b)

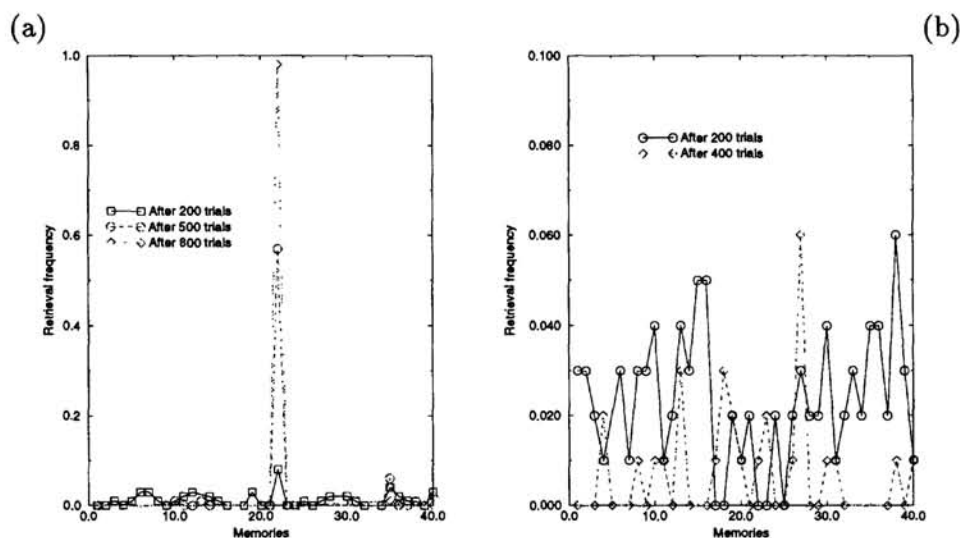

Figure 4: (a) The distribution of memory patterns spontaneously retrieved. The x-axis enumerates the memories stored, and the y-axis denotes the retrieval frequency of each memory. $\gamma = 0.0025$. (b) The distribution of stimulus-dependent retrieval of memories. $\gamma = 0.0025$.

leads to early response within days, late, delayed intervention leads to a much slower response during one or more months [11].

The current model generates some testable predictions:

- On the neuroanatomical level, the model can be tested quantitatively by searching for a positive correlation between a recent history of florid psychotic symptoms and postmortem neuropathological findings of synaptic compensation. (For example, this kind of correlation, between indices of synaptic area and cognitive functioning was found in Alzheimer patients [12]).

- On the physiological level, the increased compensatory noise should manifest itself in increased spontaneous neural activity. While this prediction is obviously difficult to examine directly, EEG studies in schizophrenics show significant increase in slow-wave delta activity which may reflect increased spontaneous activity [13].

- On the clinical level, due to the formation of a large and deep basin of attraction around the memory pattern which is at the focus of spontaneous retrieval, the proposed model predicts that its retrieval (and the elucidation of the corresponding delusions or hallucinations) may be frequently triggered by various environmental cues. A recent study points in this direction [14].

## Acknowledgements

This research has been supported by a Rothschild Fellowship to Dr. Ruppin.

# References

[1] J. Reggia, R. Berndt, and L. D'Autrechy. Connectionist models in neuropsychology. In *Handbook of Neuropsychology*, volume 9. 1994, in press.

[2] E. Ruppin. Neural modeling of psychiatric disorders. *Network: Computation in Neural Systems*, 1995. Invited review paper, to appear.

[3] J.R. Stevens. Abnormal reinnervation as a basis for schizophrenia: A hypothesis. *Arch. Gen. Psychiatry*, 49:238–243, 1992.

[4] S.K. Chaturvedi and V.D. Sinha. Recurrence of hallucinations in consecutive episodes of schizophrenia and affective disorder. *Schizophrenia Research*, 3:103–106, 1990.

[5] M. Marsel Mesulam. Schizophrenia and the brain. *New England Journal of Medicine*, 322(12):842–845, 1990.

[6] P.S. Goldman and W.J.H. Nauta. Columnar distribution of cortico-cortical fibers in the frontal, association, limbic and motor cortex of the developing rhesus monkey. *Brain Res.*, 122:393–413, 1977.

[7] L. R. Squire. Memory and the hippocampus: A synthesis from findings with rats, monkeys, and humans. *Psychological Review*, 99:195–231, 1992.

[8] M.V. Tsodyks and M.V. Feigel'man. The enhanced storage capacity in neural networks with low activity level. *Europhys. Lett.*, 6:101 – 105, 1988.

[9] D. Horn and E. Ruppin. Synaptic compensation in attractor neural networks: Modeling neuropathological findings in schizophrenia. *Neural Computation*, page To appear, 1994.

[10] W.T. Carpenter and R.W. Buchanan. Schizophrenia. *New England Journal of Medicine*, 330:10, 1994.

[11] P. Seeman. Schizophrenia as a brain disease: The dopamine receptor story. *Arch. Neurol.*, 50:1093–1095, 1993.

[12] S. T. DeKosky and S.W. Scheff. Synapse loss in frontal cortex biopsies in alzheimer's disease: Correlation with cognitive severity. *Ann. Neurology*, 27(5):457–464, 1990.

[13] Y. Jin, S.G. Potkin, D. Rice, and J. Sramek et. al. Abnormal EEG responses to photic stimulation in schizophrenic patients. *Schizophrenia Bulletin*, 16(4):627–634, 1990.

[14] R.E. Hoffman and J.A. Rapaport. A psycholoinguistic study of auditory/verbal hallucinations: Preliminary findings. In David A. and Cutting J., editors, *The Neuropsychology of Schizophrenia*. Erlbaum, 1993.
